# Computational Efficiency: A Common Organizing Principle for Parallel Computer Maps and Brain Maps?

**Mark E. Nelson    James M. Bower**
Computation and Neural Systems Program
Division of Biology, 216-76
California Institute of Technology
Pasadena, CA 91125

## ABSTRACT

It is well-known that neural responses in particular brain regions are spatially organized, but no general principles have been developed that relate the structure of a brain map to the nature of the associated computation. On parallel computers, maps of a sort quite similar to brain maps arise when a computation is distributed across multiple processors. In this paper we will discuss the relationship between maps and computations on these computers and suggest how similar considerations might also apply to maps in the brain.

## 1    INTRODUCTION

A great deal of effort in experimental and theoretical neuroscience is devoted to recording and interpreting spatial patterns of neural activity. A variety of map patterns have been observed in different brain regions and, presumably, these patterns reflect something about the nature of the neural computations being carried out in these regions. To date, however, there have been no general principles for interpreting the structure of a brain map in terms of properties of the associated computation. In the field of parallel computing, analogous maps arise when a computation is distributed across multiple processors and, in this case, the relationship

between maps and computations is better understood. In this paper, we will attempt to relate some of the mapping principles from the field of parallel computing to the organization of brain maps.

## 2    MAPS ON PARALLEL COMPUTERS

The basic idea of parallel computing is to distribute the computational workload for a single task across a large number of processors (Dongarra, 1987; Fox and Messina, 1987). In principle, a parallel computer has the potential to deliver computing power equivalent to the total computing power of the processors from which it is constructed; a 100 processor machine can potentially deliver 100 times the computing power of a single processor. In practice, however, the performance that can be achieved is always less efficient than this ideal. A perfectly efficient implementation with $N$ processors would give a factor $N$ speed up in computation time; the ratio of the actual speedup $\sigma$ to the ideal speedup $N$ can serve as a measure of the efficiency $\epsilon$ of a parallel implementation.

$$\epsilon = \frac{\sigma}{N} \tag{1}$$

For a given computation, one of the factors that most influences the overall performance is the way in which the computation is mapped onto the available processors. The efficiency of any particular mapping can be analyzed in terms of two principal factors: load-balance and communication overhead. Load-balance is a measure of how uniformly the computational work load is distributed among the available processors. Communication overhead, on the other hand, is related to the cost in time of communicating information between processors.

On parallel computers, the load imbalance $\lambda$ is defined in terms of the average calculation time per processor $T_{avg}$ and the maximum calculation time required by the busiest processor $T_{max}$:

$$\lambda = \frac{T_{max} - T_{avg}}{T_{avg}} \tag{2}$$

The communication overhead $\eta$ is defined in terms of the maximum calculation time $T_{max}$ and the maximum communication time $T_{comm}$:

$$\eta = \frac{T_{comm}}{T_{max} + T_{comm}} \tag{3}$$

Assuming that the calculation and communication phases of a computation do not overlap in time, as is the case for many parallel computers, the relationship between efficiency $\epsilon$, load-imbalance $\lambda$, and communication overhead $\eta$ is given by (Fox et al.,1988):

$$\epsilon = \frac{1-\eta}{1+\lambda} \qquad (4)$$

When both load-imbalance $\lambda$ and communication overhead $\eta$ are small, the inefficiency is approximately the sum of the contributions from load-imbalance and communication overhead:

$$\epsilon \approx 1 - (\eta + \lambda) \qquad (5)$$

When attempting to achieve maximum performance from a parallel computer, a programmer tries to find a mapping that minimizes the combined contributions of load-imbalance and communication overhead. In some cases this is accomplished by applying simple heuristics (Fox et al., 1988), while in others it requires the explicit use of optimization techniques like simulated annealing (Kirkpatrick et al., 1983) or even artificial neural network approaches (Fox and Furmanski, 1988). In any case, the optimal tradeoff between load imbalance and communication overhead depends on certain properties of the computation itself. Thus different types of computations give rise to different kinds of optimal maps on parallel computers.

## 2.1   AN EXAMPLE

In order to illustrate how different mappings can give rise to different computational efficiencies, we will consider the simulation of a single neuron using a multicompartment modeling approach (Segev et al., 1989). In such a simulation, the model neuron is divided into a large number of compartments, each of which is assumed to be isopotential. Each compartment is represented by an equivalent electric circuit that embodies information about the local membrane properties. In order to update the voltage of an individual compartment, it is necessary to know the local properties as well as the membrane voltages of the neighboring compartments. Such a model gives rise to a system of differential equations of the following form:

$$C_m \frac{dV_i}{dt} = \sum_k g_k(V_i - E_k) + g_{i-1,i}(V_{i-1} - V_i) + g_{i+1,i}(V_{i+1} - V_i) \qquad (6)$$

where $C_m$ is the membrane capacitance, $V_i$ is the membrane voltage of compartment $i$, $g_k$ and $E_k$ are the local conductances and their reversal potentials, and $g_{i\pm1,i}$ are coupling conductances to neighboring compartments.

When carrying out such a simulation on a parallel computer, where there are more compartments than processors, each processor is assigned responsibility for updating a subset of the compartments (Nelson et al., 1989). If the compartments represent equivalent computational loads, then the load-imbalance will be proportional to the difference between the maximum and the average number of compartments per processor. If the computer processors are fully interconnected by communication channels, then the communication overhead will be proportional to the number of interprocessor messages providing the voltages of neighboring compartments. If

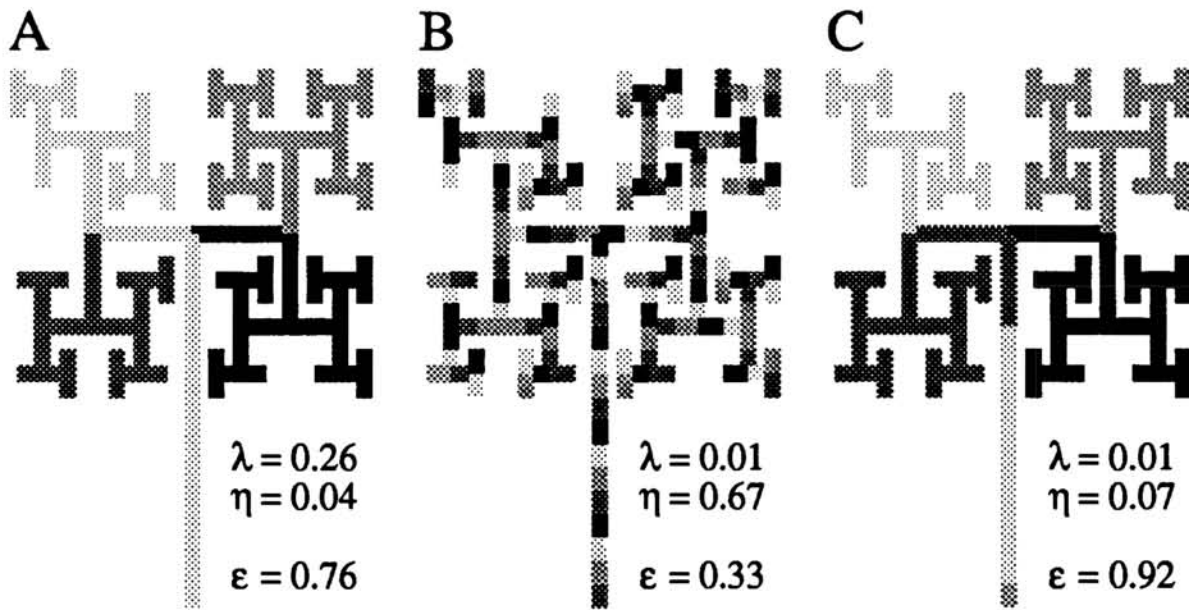

A
$\lambda = 0.26$
$\eta = 0.04$

$\varepsilon = 0.76$

B
$\lambda = 0.01$
$\eta = 0.67$

$\varepsilon = 0.33$

C
$\lambda = 0.01$
$\eta = 0.07$

$\varepsilon = 0.92$

**Figure 1:** Tradeoffs between load-imbalance $\lambda$ and communication overhead $\eta$, giving rise to different efficiencies $\epsilon$ for different mappings of a multicompartment neuron model. (A) a minimum-cut mapping that minimizes communication overhead but suffers from a significant load-imbalance, (B) a scattered mapping that minimizes load-imbalance but has a large communication overhead, and (C) a near-optimal mapping that simultaneously minimizes both load-imbalance and communication overhead.

neighboring compartments are mapped to the same processor, then this information is available without any interprocessor communication and thus no communication overhead is incurred.

Fig. 1 shows three different ways of mapping a 155 compartment neuron model onto a group of 4 processors. In each case the load-imbalance and communication overhead are calculated using the assumptions listed above and the computational efficiency is computed using eq. 4. The map in Fig. 1A minimizes the communication overhead of the mapping by making a minimum number of cuts in the dendritic tree, but is rather inefficient because a significant load-imbalance remains even after optimizing the location of each cut. The map is Fig. 1B, on the other hand, minimizes the load-imbalance, by using a scattered mapping technique (Fox et al., 1988), but is inefficient because of a large communication overhead. The map in Fig. 1C strikes a balance between load-imbalance and communication overhead that results in a high computational efficiency. Thus this particular mapping makes the best use of the available computing resources for this particular computational task.

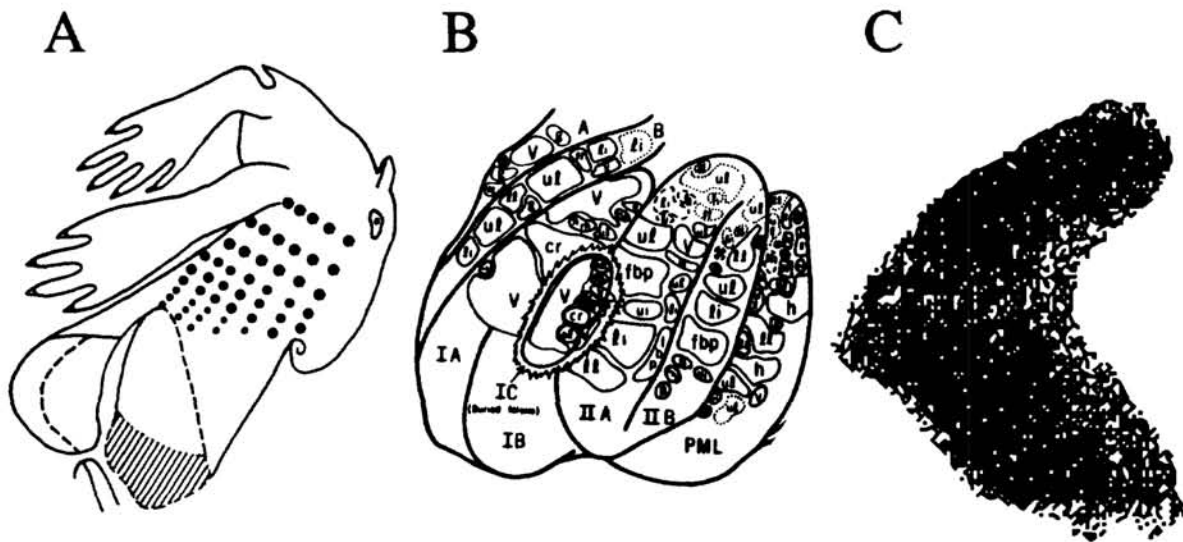

**Figure 2:** Three classes of map topologies found in the brain (of the rat). (A) continuous map of tactile inputs in somatosensory cortex (B) patchy map of tactile inputs to cerebellar cortex and (C) scattered mapping of olfactory inputs to olfactory cortex as represented by the unstructured pattern of 2DG uptake in a single section of this cortex.

## 3    MAPS IN THE BRAIN

Since some parallel computer maps are clearly more efficient than others for particular problems, it seems natural to ask whether a similar relationship might hold for brain maps and neural computations. Namely, for a given computational task, does one particular brain map topology make more efficient use of the available neural computing resources than another? If so, does this impose a significant constraint on the evolution and development of brain map topologies?

It turns out that there are striking similarities between the kinds of maps that arise on parallel computers and the types of maps that have been observed in the brain. In both cases, the map patterns can be broadly grouped into three categories: continuous maps, patchy maps, and scattered (non-topographic) maps. Fig. 2 shows examples of brain maps that fall into these categories. Fig. 2A shows an example of a smooth and continuous map representing the pattern of afferent tactile projections to the primary somatosensory cortex of a rat (Welker, 1971). The patchy map in Fig. 2B represents the spatial pattern of tactile projections to the granule cell layer of the rat cerebellar hemispheres (Shambes et al., 1978; Bower and Woolston, 1983). Finally, Fig. 2C represents an extreme case in which a brain region shows no apparent topographic organization. This figure shows the pattern of metabolic activity in one section of the olfactory (piriform) cortex, as assayed by 2-deoxyglucose (2DG) uptake, in response to the presentation of a particular odor (Sharp et al., 1977). As suggested by the uniform label in the cortex, no discernible

odor-specific patterns are found in this region of cortex.

On parallel computers, maps in these different categories arise as optimal solutions to different classes of computations. Continuous maps are optimal for computations that are local in the problem space, patchy maps are optimal for computations that involve a mixture of local and non-local interactions, and scattered maps are optimal or near-optimal for computations characterized by a high degree of interaction throughout the problem space, especially if the patterns of interaction are dynamic or cannot be easily predicted. Interestingly, it turns out that the intrinsic neural circuitry associated with different kinds of brain maps also reflects these same patterns of interaction. Brain regions with continuous maps, like somatosensory cortex, tend to have predominantly local circuitry; regions with patchy maps, like cerebellar cortex, tend to have a mixture of local and non-local circuitry; and regions with scattered maps, like olfactory cortex, tend to be characterized by wide-spread connectivity.

The apparent correspondence between brain maps and computer maps raises the general question of whether or not there are correlates of load-imbalance and communication overhead in the nervous system. In general, these factors are much more difficult to identify and quantify in the brain than on parallel computers. Parallel computer systems are, after all, human-engineered while the nervous system has evolved under a set of selection criteria and constraints that we know very little about. Furthermore, fundamental differences in the organization of digital computers and brains make it difficult to translate ideas from parallel computing directly into neural equivalents (c.f. Nelson et al., 1989). For example, it is far from clear what should be taken as the neural equivalent of a single processor. Depending on the level of analysis, it might be a localized region of a dendrite, an entire neuron, or an assembly of many neurons. Thus, one must consider multiple levels of processing in the nervous system when trying to draw analogies with parallel computers.

First we will consider the issue of load-balancing in the brain. The map in Fig. 2A, while smooth and continuous, is obviously quite distorted. In particular, the regions representing the lips and whiskers are disproportionately large in comparison to the rest of the body. It turns out that similar map distortions arise on parallel computers as a result of load-balancing. If different regions of the problem space require more computation than other regions, load-balance is achieved by distorting the map until each processor ends up with an equal share of the workload (Fox et al., 1988). In brain maps, such distortions are most often explained by variations in the density of peripheral receptors. However, it has recently been shown in the monkey, that prolonged increased use of a particular finger is accompanied by an expansion of the corresponding region of the map in the somatosensory cortex (Merzenich, 1987). Presumably this is not a consequence of a change in peripheral receptor density, but instead reflects a use-dependent remapping of some tactile computation onto available cortical circuitry.

Although such map reorganization phenomena are suggestive of load-balancing effects, we cannot push the analogy too far because we do not know what actually

corresponds to "computational load" in the brain. One possibility is that it is associated with the metabolic load that arises in response to neural activity (Yarowsky and Ingvar, 1981). Since metabolic activity necessitates the delivery of an adequate supply of oxygen and glucose via a network of small capillaries, the efficient use of the capillary system might favor mappings that tend to avoid metabolic "hot spots" which would overload the delivery capabilities of the system.

When discussing communication overhead in the brain, we also run into the problem of not knowing exactly what corresponds to "communication cost". On parallel computers, communication overhead is usually associated with the time-cost of exchanging information between processors. In the nervous system, the importance of such time-costs is probably quite dependent on the behavioral context of the computation. There is evidence, for example, that some brain regions actually make use of transmission delays to process information (Carr and Konishi, 1988). However, there is another aspect of communication overhead that may be more generally applicable having to do with the space-costs of physically connecting processors together. In the design of modern parallel computers and in the design of individual computer processor chips, space-costs associated with interconnections pose a very serious constraint for the design engineer. In the nervous system, the extremely large numbers of potential connections combined with rather strict limitations on cranial capacity are likely to make space-costs a very important factor.

## 4    CONCLUSIONS

The view that computational efficiency is an important constraint on the organization of brain maps provides a potentially useful new perspective for interpreting the structure of those maps. Although the available evidence is largely circumstantial, it seems likely that the topology of a brain map affects the efficiency with which neural resources are utilized. Furthermore, it seems reasonable to assume that network efficiency would impose a constraint on the evolution and development of map topologies that would tend to favor maps that are near-optimal for the computational tasks being performed. The very substantial task before us, in the case of the nervous system, is to carry out further experiments to better understand the detailed relationships between brain maps, neural architectures and associated computations (Bower, 1990).

### Acknowledgements

We would like to acknowledge Wojtek Furmanski and Geoffrey Fox of the Caltech Concurrent Computation Program (CCCP) for their parallel computing support. We would also like to thank Geoffrey for his comments on an earlier version of this manuscript. This effort was supported by the NSF (ECS-8700064), the Lockheed Corporation, and the Department of Energy (DE-FG03-85ER25009).

### References

Bower, J.M. (1990) Reverse engineering the nervous system: An anatomical, physiological, and computer based approach. In: *An Introduction to Neural and Electronic*

*Networks.* (S. Zornetzer, J. Davis, and C. Lau, eds), pp. 3-24, Academic Press.

Bower, J.M. and D.C. Woolston (1983) Congruence of Spatial Organization of Tactile Projections to Granule Cell and Purkinje Cell Layers of Cerebellar Hemispheres of the Albino Rat: Vertical Organization of Cerebellar Cortex. *J. Neurophysiol.* **49**, 745-756.

Carr, C.E. and M. Konishi (1988) Axonal delay lines for time measurement in the owl's brain stem. *Proc Natl Acad Sci USA* **85**, 8311-8315.

Dongarra, J.J. (1987) *Experimental Parallel Computing Architectures*, (Dongarra, J.J., ed.) North-Holland.

Fox, G. C., M. Johnson, G. Lyzenga, S. Otto, J. Salmon, D. Walker (1988) *Solving Problems on Concurrent Processors*, Prentice Hall.

Fox, G.C. and W. Furmanski (1988) Load Balancing loosely synchronous problems with a neural network. In: *Proceedings of the Third Conference on Hypercube Concurrent Computers and Applications*, (Fox, G.C., ed.), pp.241-278, ACM.

Fox, G.C. and P. Messina (1987) Advanced Computer Architectures. *Scientific American*, October, 66-74.

Kirkpatrick, S., C.D. Gelatt and M.P. Vecchi (1983) Optimization by Simulated Annealing. *Science*, **220**, 671-680.

Merzenich, M.M. (1987) Dynamic neocortical processes and the origins of higher brain functions. In: *The Neural and Molecular Bases of Learning*, (Changeux, J.-P. and Konishi, M., eds.), pp. 337-358, John Wiley & Sons.

Nelson, M.E., W. Furmanski and J.M. Bower (1989) Modeling Neurons and Networks on Parallel Computers. In: *Methods in Neuronal Modeling: From Synapses to Networks*, (Koch, C. and I. Segev, eds.), pp. 397-438, MIT Press.

Segev, I., J.W. Fleshman and R.E. Burke (1989) Compartmental Models of Complex Neurons. In: *Methods in Neuronal Modeling: From Synapses to Networks*, (Koch, C. and I. Segev, eds.), pp. 63-96, MIT Press.

Shambes, G.M., J.M. Gibson and W. Welker (1978) Fractured Somatotopy in Granule Cell Tactile Areas of Rat Cerebellar Hemispheres Revealed by Micromapping. *Brain Behav. Evol.* **15**, 94-140.

Sharp, F.R., J.S. Kauer and G.M. Shepherd (1977) Laminar Analysis of 2-Deoxyglucose Uptake in Olfactory Bulb and Olfactory Cortex of Rabbit and Rat. *J. Neurophysiol.* **40**, 800-813.

Welker, C. (1971) Microelectrode delineation of fine grain somatotopic organization of SMI cerebral neocortex in albino rat. *Brain Res.* **26**, 259-275.

Yarowsky, P.J. and D.H. Ingvar (1981) Neuronal activity and energy metabolism. *Federation Proc.* **40**, 2353-2263.
